# Intrinsic Dimension Estimation Using Packing Numbers

**Balázs Kégl**
Department of Computer Science and Operations Research
University of Montreal
CP 6128 succ. Centre-Ville, Montréal, Canada H3C 3J7
kegl@iro.umontreal.ca

## Abstract

We propose a new algorithm to estimate the intrinsic dimension of data sets. The method is based on geometric properties of the data and requires neither parametric assumptions on the data generating model nor input parameters to set. The method is compared to a similar, widely-used algorithm from the same family of geometric techniques. Experiments show that our method is more robust in terms of the data generating distribution and more reliable in the presence of noise.

## 1 Introduction

High-dimensional data sets have several unfortunate properties that make them hard to analyze. The phenomenon that the computational and statistical efficiency of statistical techniques degrade rapidly with the dimension is often referred to as the "curse of dimensionality". One particular characteristic of high-dimensional spaces is that as the volumes of constant diameter neighborhoods become large, exponentially many points are needed for reliable density estimation. Another important problem is that as the data dimension grows, sophisticated data structures constructed to speed up nearest neighbor searches rapidly become inefficient.

Fortunately, most meaningful, real life data do not uniformly fill the spaces in which they are represented. Rather, the data distributions are observed to concentrate to non-linear manifolds of low *intrinsic dimension*. Several methods have been developed to find low-dimensional representations of high-dimensional data, including Principal Component Analysis (PCA), Self-Organizing Maps (SOM) [1], Multidimensional Scaling (MDS) [2], and, more recently, Local Linear Embedding (LLE) [3] and the ISOMAP algorithm [4]. Although most of these algorithms require that the intrinsic dimension of the manifold be explicitly set, there has been little effort devoted to design and analyze techniques that estimate the intrinsic dimension of data in this context.

There are two principal areas where a good estimate of the intrinsic dimension can be useful. First, as mentioned before, the estimate can be used to set input parameters of dimension reduction algorithms. Certain methods (e.g., LLE and the ISOMAP algorithm) also require a scale parameter that determines the size of the local neighborhoods used in the algorithms. In this case, it is useful if the dimension estimate is provided as a function of the scale (see Figure 1 for an intuitive example where the intrinsic dimension of the data

depends on the resolution). Nearest neighbor searching algorithms can also profit from a good dimension estimate. The complexity of search data structures (e.g., kd-trees and R-trees) increase exponentially with the dimension, and these methods become inefficient if the dimension is more than about 20. Nevertheless, it was shown by Chávez et al. [5] that the complexity increases with the intrinsic dimension of the data rather then with the dimension of the embedding space.

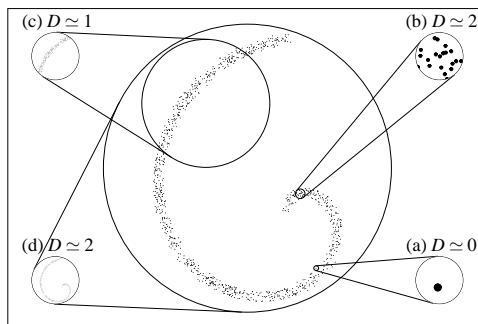

Figure 1: Intrinsic dimension $D$ at different resolutions. (a) At very small scale the data looks zero-dimensional. (b) If the scale is comparable to the noise level, the intrinsic dimension seems larger than expected. (c) The "right" scale in terms of noise and curvature. (d) At very large scale the global dimension dominates.

In this paper we present a novel method for intrinsic dimension estimation. The estimate is based on geometric properties of the data, and requires no parameters to set. Experimental results on both artificial and real data show that the algorithm is able to capture the scale dependence of the intrinsic dimension. The main advantage of the method over existing techniques is its robustness in terms of the generating distribution. The paper is organized as follows. In Section 2 we introduce the field of intrinsic dimension estimation, and give a short overview of existing approaches. The proposed algorithm is described in Section 3. Experimental results are given in Section 4.

## 2 Intrinsic dimension estimation

Informally, the intrinsic dimension of a random vector $X$ is usually defined as the number of "independent" parameters needed to represent $X$. Although in practice this informal notion seems to have a well-defined meaning, formally it is ambiguous due to the existence of space-filling curves. So, instead of this informal notion, we turn to the classical concept of *topological dimension*, and define the intrinsic dimension of $X$ as the topological dimension of the support of the distribution of $X$. For the definition, we need to introduce some notions. Given a topological space $\mathcal{X}$, the *covering* of a subset $\mathcal{S}$ is a collection $\mathcal{C}$ of open subsets in $\mathcal{X}$ whose union contains $\mathcal{S}$. A *refinement* of a covering $\mathcal{C}$ of $\mathcal{S}$ is another covering $\mathcal{C}'$ such that each set in $\mathcal{C}'$ is contained in some set in $\mathcal{C}$. The following definition is based on the observation that a $d$-dimensional set can be covered by open balls such that each point belongs to maximum $(d+1)$ open balls.

**Definition 1** *A subset $\mathcal{S}$ of a topological space $\mathcal{X}$ has **topological dimension** $D_{\text{top}}$ (also known as Lebesgue covering dimension) if every covering $\mathcal{C}$ of $\mathcal{S}$ has a refinement $\mathcal{C}'$ in which every point of $\mathcal{S}$ belongs to at most $(D_{\text{top}}+1)$ sets in $\mathcal{C}'$, and $D_{\text{top}}$ is the smallest such integer.*

The main technical difficulty with the topological dimension is that it is computationally difficult to estimate on a finite sample. Hence, practical methods use various other definitions of the intrinsic dimension. It is common to categorize intrinsic dimension estimating methods into two classes, *projection techniques* and *geometric approaches*.

Projection techniques explicitly construct a mapping, and usually measure the dimension by using some variants of principal component analysis. Indeed, given a set $\mathcal{S}_n =$

$\{X_1,\ldots,X_n\}, X_i \in \mathcal{X}, i = 1,\ldots,n$ of data points drawn independently from the distribution of $X$, probably the most obvious way to estimate the intrinsic dimension is by looking at the eigenstructure of the covariance matrix $\mathbf{C}$ of $\mathcal{S}_n$. In this approach, $\widehat{D}_{\text{pca}}$ is defined as the number of eigenvalues of $\mathbf{C}$ that are larger than a given threshold. The first disadvantage of the technique is the requirement of a threshold parameter that determines which eigenvalues are to discard. In addition, if the manifold is highly nonlinear, $\widehat{D}_{\text{pca}}$ will characterize the global (intrinsic) dimension of the data rather than the local dimension of the manifold. $\widehat{D}_{\text{pca}}$ will always overestimate $D_{\text{top}}$; the difference depends on the level of nonlinearity of the manifold. Finally, $\widehat{D}_{\text{pca}}$ can only be used if the covariance matrix of $\mathcal{S}_n$ can be calculated (e.g., when $\mathcal{X} = \mathbb{R}^d$). Although in Section 4 we will only consider Euclidean data sets, there are certain applications where only a distance metric $d : \mathcal{X} \times \mathcal{X} \mapsto \mathbb{R}^+ \cup \{0\}$ and the matrix of pairwise distances $\mathbf{D} = [d_{ij}] = d(\mathbf{x}_i, \mathbf{x}_j)$ are given.

Bruske and Sommer [6] present an approach to circumvent the second problem. Instead of doing PCA on the original data, they first cluster the data, then construct an optimally topology preserving map (OPTM) on the cluster centers, and finally, carry out PCA *locally* on the OPTM nodes. The advantages of the method are that it works well on non-linear data, and that it can produce dimension estimates at different resolutions. At the same time, the threshold parameter must still be set as in PCA, moreover, other parameters, such as the number of OPTM nodes, must also be decided by the user. The technique is similar in spirit to the way the dimension parameter of LLE is set in [3]. The algorithm runs in $O(n^2 d)$ time (where $n$ is the number of points and $d$ is the embedding dimension) which is slightly worse than the $O(nd\widehat{D}_{\text{pca}})$ complexity of the fast PCA algorithm of Roweis [7] when computing $\widehat{D}_{\text{pca}}$.

Another general scheme in the family of projection techniques is to turn the dimensionality reduction algorithm from an embedding technique into a probabilistic, generative model [8], and optimize the dimension as any other parameter by using cross-validation in a maximum likelihood setting. The main disadvantage of this approach is that the dimension estimate depends on the generative model and the particular algorithm, so if the model does not fit the data or if the algorithm does not work well on the particular problem, the estimate can be invalid.

The second basic approach to intrinsic dimension estimation is based on geometric properties of the data rather then projection techniques. Methods from this family usually require neither any explicit assumption on the underlying data model, nor input parameters to set. Most of the geometric methods use the *correlation dimension* from the family of *fractal dimensions* due to the computational simplicity of its estimation. The formal definition is based on the observation that in a $D$-dimensional set the number of pairs of points closer to each other than $r$ is proportional to $r^D$.

**Definition 2** *Given a finite set $\mathcal{S}_n = \{x_1,\ldots,x_n\}$ of a metric space $X$, let*

$$C_n(r) = \frac{2}{n(n-1)} \sum_{i=1}^{n} \sum_{j=i+1}^{n} I_{\{\|x_i - x_j\| < r\}}$$

*where $I_A$ is the indicator function of the event $A$. For a countable set $\mathcal{S} = \{x_1, x_2, \ldots\} \subset \mathcal{X}$, the* correlation integral *is defined as $C(r) = \lim_{n \to \infty} C_n(r)$. If the limit exists, the* **correlation dimension** *of S is defined as*

$$D_{\text{corr}} = \lim_{r \to 0} \frac{\log C(r)}{\log r}.$$

For a finite sample, the zero limit cannot be achieved so the estimation procedure usually consists of plotting $\log C(r)$ versus $\log r$ and measuring the slope $\frac{\partial \log C(r)}{\partial \log r}$ of the linear part

of the curve [9, 10, 11]. To formalize this intuitive procedure, we present the following definition.

**Definition 3** *The* **scale-dependent correlation dimension** *of a finite set* $S_n = \{x_1, \ldots, x_n\}$ *is*

$$\widehat{D}_{\mathrm{corr}}(r_1, r_2) = \frac{\log C(r_2) - \log C(r_1)}{\log r_2 - \log r_1}.$$

It is known that $D_{\mathrm{corr}} \leq D_{\mathrm{top}}$ and that $D_{\mathrm{corr}}$ approximates well $D_{\mathrm{top}}$ if the data distribution on the manifold is nearly uniform. However, using a non-uniform distribution on the same manifold, the correlation dimension can severely underestimate the topological dimension. To overcome this problem, we turn to the *capacity dimension*, which is another member of the fractal dimension family. For the formal definition, we need to introduce some more concepts. Given a metric space $X$ with distance metric $d(\cdot, \cdot)$, the *r-covering number* $N(r)$ of a set $S \subset X$ is the minimum number of open balls $B(x_0, r) = \{x \in X | d(x_0, x) < r\}$ whose union is a covering of $S$. The following definition is based on the observation that the covering number $N(r)$ of a $D$-dimensional set is proportional to $r^{-D}$.

**Definition 4** *The* **capacity dimension** *of a subset* $S$ *of a metric space* $X$ *is*

$$D_{\mathrm{cap}} = -\lim_{r \to 0} \frac{\log N(r)}{\log r}.$$

The principal advantage of $D_{\mathrm{cap}}$ over $D_{\mathrm{corr}}$ is that $D_{\mathrm{cap}}$ does not depend on the data distribution on the manifold. Moreover, if both $D_{\mathrm{cap}}$ and $D_{\mathrm{top}}$ exist (which is certainly the case in machine learning applications), it is known that the two dimensions agree. In spite of that, $D_{\mathrm{cap}}$ is usually discarded in practical approaches due to the high computational cost of its estimation. The main contribution of this paper is an efficient intrinsic dimension estimating method that *is* based on the capacity dimension. Experiments on both synthetic and real data confirm that our method is much more robust in terms of the data distribution than methods based on the correlation dimension.

## 3  Algorithm

Finding the covering number even of a finite set of data points is computationally difficult. To tackle this problem, first we redefine $D_{\mathrm{cap}}$ by using *packing numbers* rather than covering numbers. Given a metric space $X$ with distance metric $d(\cdot, \cdot)$, a set $\mathcal{V} \subset X$ is said to be *r-separated* if $d(x, y) \geq r$ for all distinct $x, y \in \mathcal{V}$. The *r-packing number* $M(r)$ of a set $S \subset X$ is defined as the maximum cardinality of an $r$-separated subset of $S$. The following proposition follows from the basic inequality between packing and covering numbers $N(r) \leq M(r) \leq N(r/2)$.

**Proposition 1** $D_{\mathrm{cap}} = -\lim_{r \to 0} \dfrac{\log M(r)}{\log r}.$

For a finite sample, the zero limit cannot be achieved so, similarly to the correlation dimension, we need to redefine the capacity dimension in a scale-dependent manner.

**Definition 5** *The* **scale-dependent capacity dimension** *of a finite set* $S_n = \{x_1, \ldots, x_n\}$ *is*

$$\widehat{D}_{\mathrm{cap}}(r_1, r_2) = -\frac{\log M(r_2) - \log M(r_1)}{\log r_2 - \log r_1}.$$

Finding $M(r)$ for a data set $\mathcal{S}_n = \{x_1, \ldots, x_n\}$ is equivalent to finding the cardinality of a maximum independent vertex set $\mathrm{MI}(G_r)$ of the graph $G_r(V,E)$ with vertex set $V = \mathcal{S}_n$ and edge set $E = \{(x_i, x_j) | d(x_i, x_j) < r\}$. This problem is known to be NP-hard. There are results that show that for a general graph, even the approximation of $\mathrm{MI}(G)$ within a factor of $n^{1-\varepsilon}$, for any $\varepsilon > 0$, is NP-hard [12]. On the positive side, it was shown that for such geometric graphs as $G_r$, $\mathrm{MI}(G)$ can be approximated arbitrarily well by polynomial time algorithms [13]. However, approximating algorithms of this kind scale exponentially with the data dimension both in terms of the quality of the approximation and the running time[1] so they are of little practical use for $d > 2$. Hence, instead of using one of these algorithms, we apply the following greedy approximation technique. Given a data set $\mathcal{S}_n$, we start with an empty set of centers $\mathcal{C}$, and in an iteration over $\mathcal{S}_n$ we add to $\mathcal{C}$ data points that are at a distance of at least $r$ from all the centers in $\mathcal{C}$ (lines 4 to 10 in Figure 2). The estimate $\widehat{M}(r)$ is the cardinality of $\mathcal{C}$ after every point in $\mathcal{S}_n$ has been visited.

The procedure is designed to produce an $r$-packing but certainly underestimates the packing number of the manifold, first, because we are using a finite sample, and second, because in general $\widehat{M}(r) < M(r)$. Nevertheless, we can still obtain a good estimate for $\widehat{D}_{\mathrm{cap}}$ by using $\widehat{M}(r)$ in the place of $M(r)$ in Definition 5. To see why, observe that, for a good estimate for $\widehat{D}_{\mathrm{cap}}$, it is enough if we can estimate $M(r)$ with a constant multiplicative bias independent of $r$. Although we have no formal proof that the bias of $\widehat{M}(r)$ does not change with $r$, the simple greedy procedure described above seems to work well in practice.

Even though the bias of $\widehat{M}(r)$ does not affect the estimation of $\widehat{D}_{\mathrm{cap}}$ as long as it does not change with $r$, the variance of $\widehat{M}(r)$ can distort the dimension estimate. The main source of the variance is the dependence of $\widehat{M}(r)$ on the the order of the data points in which they are visited. To eliminate this variance, we repeat the procedure several times on random permutations of the data, and compute the estimate $\widehat{D}_{\mathrm{pack}}$ by using the average of the logarithms of the packing numbers. The number of repetitions depends on $r_1$, $r_2$, and a preset parameter that determines the accuracy of the final estimate (set to 99% in all experiments) . The complete algorithm is given formally in Figure 2.

The running time of the algorithm is $O(nM(r)d)$ where $r = \min(r_1, r_2)$. At smaller scales, where $M(r)$ is comparable with $n$, it is $O(n^2d)$. On the other hand, since the variance of the estimate also tends to be smaller at smaller scales, the algorithm iterates less for the same accuracy.

## 4    Experiments

The two main objectives of the four experiments described here is to demonstrate the ability of the method to capture the scale-dependent behavior of the intrinsic dimension, and to underline its robustness in terms of the data generating distribution. In all experiments, the estimate $\widehat{D}_{\mathrm{pack}}$ is compared to the correlation dimension estimate $\widehat{D}_{\mathrm{corr}}$. Both dimensions are measured on consecutive pairs of a sequence $r_1, \ldots, r_m$ of resolutions, and the estimate is plotted halfway between the two parameters (i.e., $\widehat{D}(r_i, r_{i+1})$ is plotted at $(r_i + r_{i+1})/2$.) In the first three experiments the manifold is either known or can be approximated easily. In these experiments we use a two-sided multivariate power distribution with density

$$p(\mathbf{x}) = I_{\{\mathbf{x} \in [-1,1]^d\}} \left(\frac{p}{2}\right)^d \prod_{i=1}^{d} \left(1 - |x^{(i)}|\right)^{p-1} \tag{1}$$

```
PACKINGDIMENSION($\mathcal{S}_n, r_1, r_2, \varepsilon$)
1       for $\ell \leftarrow 1$ to $\infty$ do
2           Permute $\mathcal{S}_n$ randomly
3           for $k \leftarrow 1$ to 2 do
4               $\mathcal{C} \leftarrow \emptyset$
5               for $i \leftarrow 1$ to $n$ do
6                   for $j \leftarrow 1$ to $|\mathcal{C}|$ do
7                       if $d\big(\mathcal{S}_n[i], C[j]\big) < r_k$ then
8                           $j \leftarrow n+1$
9                   if $j < n+1$ then
10                      $\mathcal{C} \leftarrow \mathcal{C} \cup \{\mathcal{S}_n[i]\}$
11              $\widehat{L}_k[\ell] = \log|\mathcal{C}|$
12          $\widehat{D}_{\text{pack}} = -\dfrac{\mu(\widehat{L}_2) - \mu(\widehat{L}_1)}{\log r_2 - \log r_1}$
13          if $\ell > 10$ and $1.65\dfrac{\sqrt{\sigma^2(\widehat{L}_1)+\sigma^2(\widehat{L}_2)}}{\sqrt{\ell}(\log r_2 - \log r_1)} < \widehat{D}_{\text{pack}} * (1-\varepsilon)/2$ then
14              return $\widehat{D}_{\text{pack}}$
```

Figure 2: The algorithm returns the packing dimension estimate $\widehat{D}_{\text{pack}}(r_1, r_2)$ of a data set $\mathcal{S}_n$ with $\varepsilon$ accuracy nine times out of ten.

with different exponents $p$ to generate uniform ($p = 1$) and non-uniform data sets on the manifold.

The first synthetic data is that of Figure 1. We generated 5000 points on a spiral-shaped manifold with a small uniform perpendicular noise. The curves in Figure 3(a) reflect the scale-dependency observed in Figure 1. As the distribution becomes uneven, $\widehat{D}_{\text{corr}}$ severely underestimates $\widehat{D}_{\text{top}}$ while $\widehat{D}_{\text{pack}}$ remains stable.

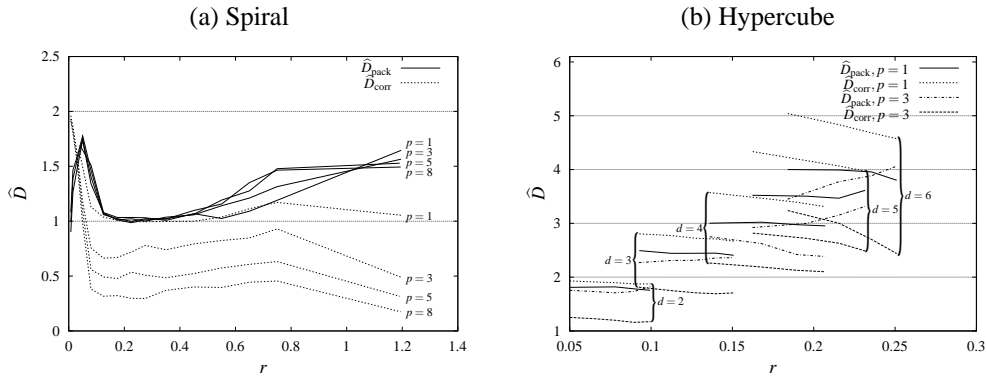

Figure 3: Intrinsic dimension of (a) a spiral-shaped manifold and (b) hypercubes of different dimensions. The curves reflect the scale-dependency observed in Figure 1. The more uneven the distribution, the more $\widehat{D}_{\text{corr}}$ underestimates $\widehat{D}_{\text{top}}$ while $\widehat{D}_{\text{pack}}$ remains relatively stable.

The second set of experiments were designed to test how well the methods estimate the dimension of 5000 data points generated in hypercubes of dimensions two to six (Figure 3(b)). In general, both $\widehat{D}_{\text{corr}}$ and $\widehat{D}_{\text{pack}}$ underestimates $\widehat{D}_{\text{top}}$. The negative bias grows with the dimension, probably due to the fact that data sets of equal cardinality become

sparser in a higher dimensional space. To compensate this bias on a general data set, we use a standard calibration procedure to correct the estimate by the bias observed on a uniformly generated data set of the same cardinality. Our experiment shows that, in the case of $D_{\text{corr}}$, this calibrating procedure can fail if the distribution is highly non-uniform.

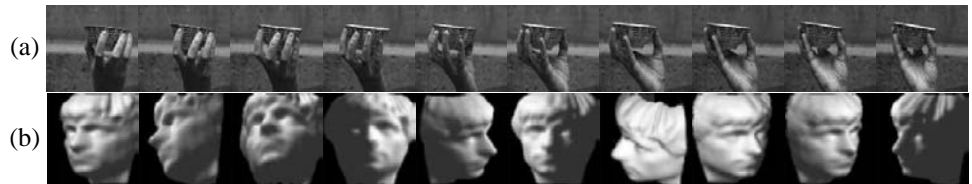

We also tested the methods on two sets of image data. Both sets contained $64 \times 64$ images with 256 gray levels. The images were normalized so that the distance between a black and a white image is 1. The first set is a sequence of 481 snapshots of a hand turning a cup from the CMU database[2] (Figure 4(a)). The sequence of images sweeps a curve in a 4096-dimensional space so its informal intrinsic dimension is one. Figure 5(a) shows that at a small scale, both methods find a local dimension between 1 and 2. At a slightly higher scale the intrinsic dimension increases indicating a relatively high curvature of the image sequence curve. To test the distribution dependence of the estimates, we constructed a polygonal curve by connecting consecutive points of the sequence, and resampled 481 points by using the power distribution (1) with $p = 2, 3$. We also constructed a highly-uniform, lattice-like data set by drawing approximately equidistant consecutive points from the polygonal curve. Our results in Figure 5(a) confirm again that $\widehat{D}_{\text{corr}}$ varies extensively with the generating distribution on the manifold while $\widehat{D}_{\text{pack}}$ remains remarkably stable.

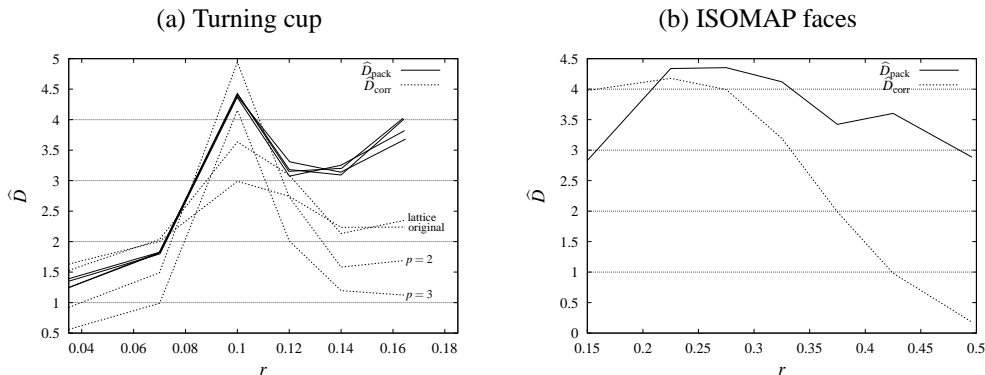

Figure 4: The real datasets. (a) Sequence of snapshots of a hand turning a cup. (b) Faces database from ISOMAP [4].

The final experiment was conducted on the "faces" database from the ISOMAP paper [4] (Figure 4(b)). The data set contained 698 images of faces generated by using three free parameters: vertical and horizontal orientation, and light direction. Figure 5(b) indicates that both estimates are reasonably close to the informal intrinsic dimension.

(a) Turning cup

(b) ISOMAP faces

Figure 5: The intrinsic dimension of image data sets.

We found in all experiments that at a very small scale $\widehat{D}_{\text{corr}}$ tends to be higher than $\widehat{D}_{\text{pack}}$,

while $\widehat{D}_{\mathrm{pack}}$ tends to be more stable as the scale grows. Hence, if the data contains very little noise and it is generated uniformly on the manifold, $\widehat{D}_{\mathrm{corr}}$ seems to be closer to the "real" intrinsic dimension. On the other hand, if the data contains noise (in which case at a very small scale we are estimating the dimension of the noise rather than the dimension of the manifold), or the distribution on the manifold is non-uniform, $\widehat{D}_{\mathrm{pack}}$ seems more reliable than $\widehat{D}_{\mathrm{corr}}$.

## 5   Conclusion

We have presented a new algorithm to estimate the intrinsic dimension of data sets. The method estimates the packing dimension of the data and requires neither parametric assumptions on the data generating model nor input parameters to set. The method is compared to a widely-used technique based on the correlation dimension. Experiments show that our method is more robust in terms of the data generating distribution and more reliable in the presence of noise.

## Footnotes

[1]Typically, the computation of an independent vertex set of $G$ of size at least $\left(1 - \frac{1}{k}\right)^d \mathrm{MI}(G)$ requires $O(n^{k^d})$ time.

[2] http://vasc.ri.cmu.edu/idb/html/motion/hand/index.html

## References

[1]  T. Kohonen, *The Self-Organizing Map*, Springer-Verlag, 2nd edition, 1997.

[2]  T. F. Cox and M. A. Cox, *Multidimensional Scaling*, Chapman & Hill, 1994.

[3]  S. Roweis and Saul L. K., "Nonlinear dimensionality reduction by locally linear embedding," *Science*, vol. 290, pp. 2323–2326, 2000.

[4]  J. B. Tenenbaum, V. de Silva, and Langford J. C., "A global geometric framework for nonlinear dimensionality reduction," *Science*, vol. 290, pp. 2319–2323, 2000.

[5]  E. Chávez, G. Navarro, R. Baeza-Yates, and J. Marroquín, "Searching in metric spaces," *ACM Computing Surveys*, p. to appear, 2001.

[6]  J. Bruske and G. Sommer, "Intrinsic dimensionality estimation with optimally topology preserving maps," *IEEE Transactions on Pattern Analysis and Machine Intelligence*, vol. 20, no. 5, pp. 572–575, 1998.

[7]  S. Roweis, "EM algorithms for PCA and SPCA," in *Advances in Neural Information Processing Systems*. 1998, vol. 10, pp. 626–632, The MIT Press.

[8]  C. M. Bishop, M. Svensén, and C. K. I. Williams, "GTM: The generative topographic mapping," *Neural Computation*, vol. 10, no. 1, pp. 215–235, 1998.

[9]  P. Grassberger and I. Procaccia, "Measuring the strangeness of strange attractors," *Physica*, vol. D9, pp. 189–208, 1983.

[10]  F. Camastra and A. Vinciarelli, "Estimating intrinsic dimension of data with a fractal-based approach," *IEEE Transactions on Pattern Analysis and Machine Intelligence*, 2002, to appear.

[11]  A. Belussi and C. Faloutsos, "Spatial join selectivity estimation using fractal concepts," *ACM Transactions on Information Systems*, vol. 16, no. 2, pp. 161–201, 1998.

[12]  J. Hastad, "Clicque is hard to approximate within $n^{1-\varepsilon}$," in *Proceedings of the 37th Annual Symposium on Foundations of Computer Science FOCS'96*, 1996, pp. 627–636.

[13]  T. Erlebach, K. Jansen, and E. Seidel, "Polynomial-time approximation schemes for geometric graphs," in *Proceedings of the 12th ACM-SIAM Symposium on Discrete Algorithms SODA'01*, 2001, pp. 671–679.
